# Learning to Hash with Binary Reconstructive Embeddings

**Brian Kulis and Trevor Darrell**
UC Berkeley EECS and ICSI
Berkeley, CA
{kulis,trevor}@eecs.berkeley.edu

## Abstract

Fast retrieval methods are increasingly critical for many large-scale analysis tasks, and there have been several recent methods that attempt to learn hash functions for fast and accurate nearest neighbor searches. In this paper, we develop an algorithm for learning hash functions based on explicitly minimizing the reconstruction error between the original distances and the Hamming distances of the corresponding binary embeddings. We develop a scalable coordinate-descent algorithm for our proposed hashing objective that is able to efficiently learn hash functions in a variety of settings. Unlike existing methods such as semantic hashing and spectral hashing, our method is easily kernelized and does not require restrictive assumptions about the underlying distribution of the data. We present results over several domains to demonstrate that our method outperforms existing state-of-the-art techniques.

## 1 Introduction

Algorithms for fast indexing and search have become important for a variety of problems, particularly in the domains of computer vision, text mining, and web databases. In cases where the amount of data is huge—large image repositories, video sequences, and others—having fast techniques for finding nearest neighbors to a query is essential. At an abstract level, we may view hashing methods for similarity search as mapping input data (which may be arbitrarily high-dimensional) to a low-dimensional binary (Hamming) space. Unlike standard dimensionality-reduction techniques from machine learning, the fact that the embeddings are binary is critical to ensure fast retrieval times—one can perform efficient linear scans of the binary data to find the exact nearest neighbors in the Hamming space, or one can use data structures for finding approximate nearest neighbors in the Hamming space which have running times that are sublinear in the number of total objects [1, 2]. Since the Hamming distance between two objects can be computed via an xor operation and a bit count, even a linear scan in the Hamming space for a nearest neighbor to a query in a database of 100 million objects can currently be performed within a few seconds on a typical workstation. If the input dimensionality is very high, hashing methods lead to enormous computational savings.

In order to be successful, hashing techniques must appropriately preserve distances when mapping to the Hamming space. One of the basic but most widely-employed methods, locality-sensitive hashing (LSH) [1, 2], generates embeddings via random projections and has been used for many large-scale search tasks. An advantage to this technique is that the random projections provably maintain the input distances in the limit as the number of hash bits increases; at the same time, it has been observed that the number of hash bits required may be large in some cases to faithfully maintain the distances. On the other hand, several recent techniques—most notably semantic hashing [3] and spectral hashing [4]—attempt to overcome this problem by designing hashing techniques that leverage machine learning to find appropriate hash functions to optimize an underlying hashing objective. Both methods have shown advantages over LSH in terms of the number of bits required

to find good approximate nearest neighbors. However, these methods cannot be directly applied in kernel space and have assumptions about the underlying distributions of the data. In particular, as noted by the authors, spectral hashing assumes a uniform distribution over the data, a potentially restrictive assumption in some cases.

In this paper, we introduce and analyze a simple objective for learning hash functions, develop an efficient coordinate-descent algorithm, and demonstrate that the proposed approach leads to improved results as compared to existing hashing techniques. The main idea is to construct hash functions that explicitly preserve the input distances when mapping to the Hamming space. To achieve this, we minimize a squared loss over the error between the input distances and the reconstructed Hamming distances. By analyzing the reconstruction objective, we show how to efficiently and exactly minimize the objective function with respect to a single variable. If there are $n$ training points, $k$ nearest neighbors per point in the training data, and $b$ bits in our desired hash table, our method ends up costing $O(nb(k + \log n))$ time per iteration to update all hash functions, and provably reaches a local optimum of the reconstruction objective. In experiments, we compare against relevant existing hashing techniques on a variety of important vision data sets, and show that our method is able to compete with or outperform state-of-the-art hashing algorithms on these data sets. We also apply our method on the very large Tiny Image data set of 80 million images [5], to qualitatively show some example retrieval results obtained by our proposed method.

## 1.1 Related Work

Methods for fast nearest neighbor retrieval are generally broken down into two families. One group partitions the data space recursively, and includes algorithms such as $k - d$ trees [6], M-trees [7], cover trees [8], metric trees [9], and other related techniques. These methods attempt to speed up nearest neighbor computation, but can degenerate to a linear scan in the worst case. Our focus in this paper is on hashing-based methods, which map the data to a low-dimensional Hamming space. Locality-sensitive hashing [1, 2] is the most popular method, and extensions have been explored for accommodating distances such as $\ell_p$ norms [10], learned metrics [11], and image kernels [12]. Algorithms based on LSH typically come with guarantees that the approximate nearest neighbors (neighbors within $(1 + \epsilon)$ times the true nearest neighbor distance) may be found in time that is sublinear in the total number of database objects (but as a function of $\epsilon$). Unlike standard dimensionality-reduction techniques, the binary embeddings allow for extremely fast similarity search operations. Several recent methods have explored ways to improve upon the random projection techniques used in LSH. These include semantic hashing [3], spectral hashing [4], parameter-sensitive hashing [13], and boosting-based hashing methods [14].

## 2 Hashing Formulation

In the following section, we describe our proposed method, starting with the choice of parameterization for the hash functions and the objective function to minimize. We then develop a coordinate-descent algorithm used to minimize the objective function, and discuss extensions of the proposed approach.

### 2.1 Setup

Let our data set be represented by a set of $n$ vectors, given by $X = [\boldsymbol{x}_1 \; \boldsymbol{x}_2 \; ... \; \boldsymbol{x}_n]$. We will assume that these vectors are normalized to have unit $\ell_2$ norm—this will make it easier to maintain the proper scale for comparing distances in the input space to distance in the Hamming space.[1] Let a kernel function over the data be denoted as $\kappa(\boldsymbol{x}_i, \boldsymbol{x}_j)$. We use a kernel function as opposed to the standard inner product to emphasize that the algorithm can be expressed purely in kernel form.

We would like to project each data point to a low-dimensional binary space to take advantage of fast nearest neighbor routines. Suppose that the desired number of dimensions of the binary space is $b$; we will compute the $b$-dimensional binary embedding by projecting our data using a set of $b$ hash functions $h_1, ..., h_b$. Each hash function $h_i$ is a binary-valued function, and our low-dimensional

binary reconstruction can be represented as $\tilde{\boldsymbol{x}}_i = [h_1(\boldsymbol{x}_i); h_2(\boldsymbol{x}_i); ...; h_b(\boldsymbol{x}_i)]$. Finally, denote $d(\boldsymbol{x}_i, \boldsymbol{x}_j) = \frac{1}{2}\|\boldsymbol{x}_i - \boldsymbol{x}_j\|^2$ and $\tilde{d}(\boldsymbol{x}_i, \boldsymbol{x}_j) = \frac{1}{b}\|\tilde{\boldsymbol{x}}_i - \tilde{\boldsymbol{x}}_j\|^2$. Notice that $d$ and $\tilde{d}$ are always between 0 and 1.

## 2.2 Parameterization and Objective

In standard random hyperplane locality-sensitive hashing (e.g. [1]), each hash function $h_p$ is generated independently by selecting a random vector $\boldsymbol{r}_p$ from a multivariate Gaussian with zero-mean and identity covariance. Then the hash function is given as $h_p(\boldsymbol{x}) = \text{sign}(\boldsymbol{r}_p^T \boldsymbol{x})$. In contrast, we propose to generate a sequence of hash functions that are dependent on one another, in the same spirit as in spectral hashing (though with a different parameterization). We introduce a matrix $W$ of size $b \times n$, and we parameterize the hash functions $h_1, ..., h_p, ..., h_b$ as follows:

$$h_p(\boldsymbol{x}) = \text{sign}\left(\sum_{q=1}^{s} W_{pq}\kappa(\boldsymbol{x}_{pq}, \boldsymbol{x})\right).$$

Note that the data points $\boldsymbol{x}_{pq}$ for each hash function need not be the same for each $h_q$ (that is, each hash function may utilize different sets of points). Similarly, the number of points $s$ used for each hash function may change, though for simplicity we will present the case when $s$ is the same for each function (and so we can represent all weights via the $b \times s$ matrix $W$). Though we are not aware of any existing methods that parameterize the hash functions in this way, this parameterization is natural for several reasons. It does not explicitly assume anything about the distribution of the data. It is expressed in kernelized form, meaning we can easily work over a variety of input data. Furthermore, the form of each hash function—the sign of a linear combination of kernel function values—is the same as several kernel-based learning algorithms such as support vector machines.

Rather than simply choosing the matrix $W$ based on random hyperplanes, we will specifically *construct* this matrix to achieve good reconstructions. In particular, we will look at the squared error between the original distances (using $d$) and the reconstructed distances (using $\tilde{d}$). We minimize the following objective with respect to the weight matrix $W$:

$$\mathcal{O}(\{\boldsymbol{x}_i\}_{i=1}^n, W) = \sum_{(i,j)\in\mathcal{N}} (d(\boldsymbol{x}_i, \boldsymbol{x}_j) - \tilde{d}(\boldsymbol{x}_i, \boldsymbol{x}_j))^2. \tag{1}$$

The set $\mathcal{N}$ is a selection of pairs of points, and can be chosen based on the application. Typically, we will choose this to be a set of pairs which includes both the nearest neighbors as well as other pairs from the database (see Section 3 for details). If we choose $k$ pairs for each point, then the total size of $\mathcal{N}$ will be $nk$.

## 2.3 Coordinate-Descent Algorithm

The objective $\mathcal{O}$ given in (1) is highly non-convex in $W$, making optimization the main challenge in using the proposed objective for hashing. One of the most difficult issues is due to the fact that the reconstructions are binary; the objective is not continuous or differentiable, so it is not immediately clear how an effective algorithm would proceed. One approach is to replace the sign function by the sigmoid function, as is done with neural networks and logistic regression.[2] Then the objective $\mathcal{O}$ and gradient $\nabla\mathcal{O}$ can both be computed in $O(nkb)$ time. However, our experience with minimizing $\mathcal{O}$ with such an approach using a quasi-Newton L-BFGS algorithm typically resulted in poor local optima; we need an alternative method.

Instead of the continuous relaxation, we will consider fixing all but one weight $W_{pq}$, and optimize the original objective $\mathcal{O}$ with respect to $W_{pq}$. Surprisingly, we will show below that an *exact*, optimal update to this weight can be achieved in time $O(n \log n + nk)$. Such an approach will update a single hash function $h_p$; then, by choosing a single weight to update for each hash function, we can update all hash functions in $O(nb(k + \log n))$ time. In particular, if $k = \Omega(\log n)$, then we can update all hash functions on the order of the time it takes to compute the objective function itself, making the updates particularly efficient. We will also show that this method provably converges to a local optimum of the objective function $\mathcal{O}$.

We sketch out the details of our coordinate-descent scheme below. We begin with a simple lemma characterizing how the objective function changes when we update a single hash function.

**Lemma 1.** *Let* $\bar{D}_{ij} = d(\boldsymbol{x}_i, \boldsymbol{x}_j) - \tilde{d}(\boldsymbol{x}_i, \boldsymbol{x}_j)$. *Consider updating some hash function* $h_{old}$ *to* $h_{new}$ *(where* $\tilde{d}$ *uses* $h_{old}$*), and let* $\boldsymbol{h}_o$ *and* $\boldsymbol{h}_n$ *be the* $n \times 1$ *vectors obtained by applying the old and new hash functions to each data point, respectively. Then the objective function* $\mathcal{O}$ *from* (1) *after updating the hash function can be expressed as*

$$\mathcal{O} = \sum_{(i,j) \in \mathcal{N}} \left( \bar{D}_{ij} + \frac{1}{b}(\boldsymbol{h}_o(i) - \boldsymbol{h}_o(j))^2 - \frac{1}{b}(\boldsymbol{h}_n(i) - \boldsymbol{h}_n(j))^2 \right)^2.$$

*Proof.* For notational convenience in this proof, let $\tilde{D}_{old}$ and $\tilde{D}_{new}$ be the matrices of reconstructed distances using $h_{old}$ and $h_{new}$, respectively, and let $H_{old}$ and $H_{new}$ be the $n \times b$ matrices of old and new hash bits, respectively. Also, let $\boldsymbol{e}_t$ be the $t$-th standard basis vector and $\boldsymbol{e}$ be a vector of all ones. Note that $H_{new} = H_{old} + (\boldsymbol{h}_n - \boldsymbol{h}_o)\boldsymbol{e}_t^T$, where $t$ is the index of the hash function being updated. We can express $\tilde{D}_{old}$ as

$$\tilde{D}_{old} = \frac{1}{b}\left( \boldsymbol{\ell}_{old}\boldsymbol{e}^T + \boldsymbol{e}\boldsymbol{\ell}_{old}^T - 2H_{old}H_{old}^T \right),$$

where $\boldsymbol{\ell}_{old}$ is the vector of squared norms of the rows of $H_{old}$. Note that the corresponding vector of squared norms of the rows of $H_{new}$ may be expressed as $\boldsymbol{\ell}_{new} = \boldsymbol{\ell}_{old} - \boldsymbol{h}_o + \boldsymbol{h}_n$ since the hash vectors are binary-valued. Therefore we may write

$$
\begin{aligned}
\tilde{D}_{new} &= \frac{1}{b}\Big( (\boldsymbol{\ell}_{old} + \boldsymbol{h}_n - \boldsymbol{h}_o)\boldsymbol{e}^T + \boldsymbol{e}(\boldsymbol{\ell}_{old} + \boldsymbol{h}_n - \boldsymbol{h}_o)^T \\
&\qquad -2(H_{old} + (\boldsymbol{h}_n - \boldsymbol{h}_o)\boldsymbol{e}_t^T)(H_{old} + (\boldsymbol{h}_n - \boldsymbol{h}_o)\boldsymbol{e}_t^T)^T \Big) \\
&= \tilde{D}_{old} + \frac{1}{b}\Big( (\boldsymbol{h}_n - \boldsymbol{h}_o)\boldsymbol{e}^T + \boldsymbol{e}(\boldsymbol{h}_n - \boldsymbol{h}_o)^T - 2(\boldsymbol{h}_n\boldsymbol{h}_n^T - \boldsymbol{h}_o\boldsymbol{h}_o^T) \Big) \\
&= \tilde{D}_{old} - \frac{1}{b}\Big( (\boldsymbol{h}_o\boldsymbol{e}^T + \boldsymbol{e}\boldsymbol{h}_o^T - 2\boldsymbol{h}_o\boldsymbol{h}_o^T) - (\boldsymbol{h}_n\boldsymbol{e}^T + \boldsymbol{e}\boldsymbol{h}_n^T - 2\boldsymbol{h}_n\boldsymbol{h}_n^T) \Big),
\end{aligned}
$$

where we have used the fact that $H_{old}\boldsymbol{e}_t = \boldsymbol{h}_o$. We can then write the objective using $\tilde{D}_{new}$ to obtain

$$
\begin{aligned}
\mathcal{O} &= \sum_{(i,j) \in \mathcal{N}} \left( \bar{D}_{ij} + \frac{1}{b}(\boldsymbol{h}_o(i) + \boldsymbol{h}_o(j) - 2\boldsymbol{h}_o(i)\boldsymbol{h}_o(j)) - \frac{1}{b}(\boldsymbol{h}_n(i) + \boldsymbol{h}_n(j) - 2\boldsymbol{h}_n(i)\boldsymbol{h}_n(j)) \right)^2 \\
&= \sum_{(i,j) \in \mathcal{N}} \left( \bar{D}_{ij} + \frac{1}{b}(\boldsymbol{h}_o(i) - \boldsymbol{h}_o(j))^2 - \frac{1}{b}(\boldsymbol{h}_n(i) - \boldsymbol{h}_n(j))^2 \right)^2,
\end{aligned}
$$

since $\boldsymbol{h}_o(i)^2 = \boldsymbol{h}_o(i)$ and $\boldsymbol{h}_n(i)^2 = \boldsymbol{h}_n(i)$. This completes the proof. □

The lemma above demonstrates that, when updating a hash function, the new objective function can be computed in $O(nk)$ time, assuming that we have computed and stored the values of $\bar{D}_{ij}$. Next we show that we can compute an optimal weight update in time $O(nk + n \log n)$.

Consider choosing some hash function $h_p$, and choose one weight index $q$, i.e. fix all entries of $W$ except $W_{pq}$, which corresponds to the one weight updated during this iteration of coordinate-descent. Modifying the value of $W_{pq}$ results in updating $h_p$ to a new hashing function $h_{new}$. Now, for every point $\boldsymbol{x}$, there is a *hashing threshold*: a new value of $W_{pq}$, which we will call $\hat{W}_{pq}$, such that

$$\sum_{q=1}^{s} \hat{W}_{pq}\kappa(\boldsymbol{x}_{pq}, \boldsymbol{x}) = 0.$$

Observe that, if $c_{\boldsymbol{x}} = \sum_{q=1}^{s} W_{pq} \kappa(\boldsymbol{x}_{pq}, \boldsymbol{x})$, then the threshold $t_{\boldsymbol{x}}$ is given by

$$t_{\boldsymbol{x}} = W_{pq} - \frac{c_{\boldsymbol{x}}}{\kappa(\boldsymbol{x}_{pq}, \boldsymbol{x})}.$$

We first compute the thresholds for all $n$ data points: once we have the values of $c_{\boldsymbol{x}}$ for all $\boldsymbol{x}$, computing $t_{\boldsymbol{x}}$ for all points requires $O(n)$ time. Since we are updating a single $W_{pq}$ per iteration, we can update the values of $c_{\boldsymbol{x}}$ in $O(n)$ time after updating $W_{pq}$, so the total time to compute all thresholds $t_{\boldsymbol{x}}$ is $O(n)$.

Next, we sort the thresholds in increasing order, which defines a set of $n + 1$ intervals (interval 0 is the interval of values smaller than the first threshold, interval 1 is the interval of points between the first and the second threshold, and so on). Observe that, for any fixed interval, the new computed hash function $h_{new}$ does not change over the entire interval. Furthermore, observe that as we cross from one threshold to the next, a single bit of the corresponding hash vector flips. As a result, we need only compute the objective function at each of the $n + 1$ intervals, and choose the interval that minimizes the objective function. We choose a value $W_{pq}$ within that interval (which will be optimal) and update the hash function using this new choice of weight. The following result shows that we can choose the appropriate interval in time $O(nk)$. When we add the cost of sorting the thresholds, the total cost of an update to a single weight $W_{pq}$ is $O(nk + n \log n)$.

**Lemma 2.** *Consider updating a single hash function. Suppose we have a sequence of hash vectors $\boldsymbol{h}_{t_0}, ..., \boldsymbol{h}_{t_n}$ such that $\boldsymbol{h}_{t_{j-1}}$ and $\boldsymbol{h}_{t_j}$ differ by a single bit for $1 \leq j \leq n$. Then the objective functions for all $n + 1$ hash functions can be computed in $O(nk)$ time.*

*Proof.* The objective function may be computed in $O(nk)$ time for the hash function $\boldsymbol{h}_{t_0}$ corresponding to the smallest interval. Consider the case when going from $\boldsymbol{h}_o = \boldsymbol{h}_{t_{j-1}}$ to $\boldsymbol{h}_n = \boldsymbol{h}_{t_j}$ for some $1 \leq j \leq n$. Let the index of the bit that changes in $\boldsymbol{h}_n$ be $a$. The only terms of the sum in the objective that change are ones of the form $(a, j) \in \mathcal{N}$ and $(i, a) \in \mathcal{N}$. Let $f_a = 1$ if $\boldsymbol{h}_o(a) = 0, \boldsymbol{h}_n(a) = 1$, and $f_a = -1$ otherwise. Then we can simplify $(\boldsymbol{h}_n(i) - \boldsymbol{h}_n(j))^2 - (\boldsymbol{h}_o(i) - \boldsymbol{h}_o(j))^2$ to $f_a(1 - 2\boldsymbol{h}_n(j))$ when $a = i$ and to $f_a(1 - 2\boldsymbol{h}_n(i))$ when $a = j$ (the expression is zero when $i = j$ and will not contribute to the objective). Therefore the relevant terms in the objective function as given in Lemma 1 may be written as:

$$\sum_{(a,j) \in \mathcal{N}} \left( \bar{D}_{aj} - \frac{f_a}{b}(1 - 2\boldsymbol{h}_n(j)) \right)^2 + \sum_{(i,a) \in \mathcal{N}} \left( \bar{D}_{ia} - \frac{f_a}{b}(1 - 2\boldsymbol{h}_n(i)) \right)^2.$$

As there are $k$ nearest neighbors, the first sum will have $k$ elements and can be computed in $O(k)$ time. The second summation may have more or less than $k$ terms, but across all data points there will be $k$ terms on average. Furthermore, we must update $\bar{D}$ as we progress through the hash functions, which can also be straightforwardly done in $O(k)$ time on average. Completing this process over all $n + 1$ hash functions results in a total of $O(nk)$ time. $\qquad\square$

Putting everything together, we have shown the following result:

**Theorem 3.** *Fix all but one entry $W_{pq}$ of the hashing weight matrix $W$. An optimal update to $W_{pq}$ to minimize* (1) *may be computed in $O(nk + n \log n)$ time.*

Our overall strategy successively cycles through each hash function one by one, randomly selects a weight to update for each hash function, and computes the optimal updates for those weights. It then repeats this process until reaching local convergence. One full iteration to update all hash functions requires time $O(nb(k + \log n))$. Note that local convergence is guaranteed in a finite number of updates since each update will never increase the objective function value, and only a finite number of possible hash configurations are possible.

## 2.4 Extensions

The method described in the previous section may be enhanced in various ways. For instance, the algorithm we developed is completely unsupervised. One could easily extend the method to a supervised one, which would be useful for example in large-scale $k$-NN classification tasks. In this scenario, one would additionally receive a set of similar and dissimilar pairs of points based on

class labels or other background knowledge. For all similar pairs, one could set the target original distance to be zero, and for all dissimilar pairs, one could set the target original distance to be large (say, 1).

One may also consider loss functions other than the quadratic loss considered in this paper. Another option would be to use an $\ell_1$-type loss, which would not penalize outliers as severely. Additionally, one may want to introduce regularization, especially for the supervised case. For example, the addition of an $\ell_1$ regularization over the entries of $W$ could lead to sparse hash functions, and may be worth additional study.

## 3    Experiments

We now present results comparing our proposed approach to the relevant existing methods—locality sensitive hashing, semantic hashing (RBM), and spectral hashing. We also compared against the Boosting SSC algorithm [14] but were unable to find parameters to yield competitive performance, and so we do not present those results here. We implemented our binary reconstructive embedding method (BRE) and LSH, and used the same code for spectral hashing and RBM that was employed in [4]. We further present some qualitative results over the Tiny Image data set to show example retrieval results obtained by our method.

### 3.1    Data Sets and Methodology

We applied the hashing algorithms to a number of important large-scale data sets from the computer vision community. Our vision data sets include: the Photo Tourism data [15], a collection of approximately 300,000 image patches, processed using SIFT to form 128-dimensional vectors; the Caltech-101 [16], a standard benchmark for object recognition in the vision community; and LabelMe and Peekaboom [17], two image data set on top of which global Gist descriptors have been extracted. We also applied our method to MNIST, the standard handwritten digits data set, and Nursery, one of the larger UCI data sets.

We mean-centered the data and normalized the feature vectors to have unit norm. Following the suggestion in [4], we apply PCA (or kernel PCA in the case of kernelized data) to the input data before applying spectral hashing or BRE—the results of the RBM method and LSH were better without applying PCA, so PCA is not applied for these algorithms. For all data sets, we trained the methods using 1000 randomly selected data points. For training the BRE method, we select nearest neighbors using the top 5th percentile of the training distances and set the target distances to 0; we found that this ensures that the nearest neighbors in the embedded space will have Hamming distance very close to 0. We also choose farthest neighbors using the 98th percentile of the training distances and maintained their original distances as target distances. Having both near and far neighbors improves performance for BRE, as it prevents a trivial solution where all the database objects are given the same hash key. The spectral hashing and RBM parameters are set as in [4, 17]. After constructing the hash functions for each method, we randomly generate 3000 hashing queries (except for Caltech-101, which has fewer than 4000 data points; in this case we choose the remainder of the data as queries).

We follow the evaluation scheme developed in [4]. We collect training/test pairs such that the unnormalized Hamming distance using the constructed hash functions is less than or equal to three. We then compute the percentage of these pairs that are nearest neighbors in the original data space, which are defined as pairs of points from the training set whose distances are in the top 5th percentile. This percentage is plotted as the number of bits increases. Once the number of bits is sufficiently high (e.g. 50), one would expect that distances with a Hamming distance less than or equal to three would correspond to nearest neighbors in the original data embedding.

### 3.2    Quantitative Results

In Figure 1, we plot hashing retrieval results over each of the data sets. We can see that the BRE method performs comparably to or outperforms the other methods on all data sets. Observe that both RBM and spectral hashing underperform all other methods on at least one data set. On some

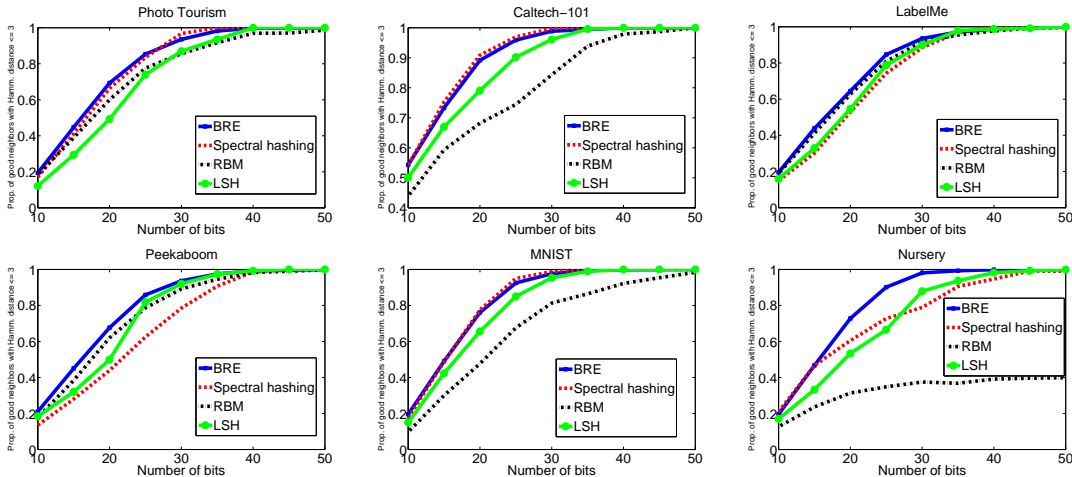

Figure 1: Results over Photo Tourism, Caltech-101, LabelMe, Peekaboom, MNIST, and Nursery. The plots show how well the nearest neighbors in the Hamming space (pairs of data points with unnormalized Hamming distance less than or equal to 3) correspond to the nearest neighbors (top 5th percentile of distances) in the original dataset. Overall, our method outperforms, or performs comparably to, existing methods. See text for further details.

data sets, RBM appears to require significantly more than 1000 training images to achieve good performance, and in these cases the training time is substantially higher than the other methods.

One surprising outcome of these results is that LSH performs well in comparison to the other existing methods (and outperforms some of them for some data sets)—this stands in contrast to the results of [4], where LSH showed significantly poorer performance (we also evaluated our LSH implementation using the same training/test split as in [4] and found similar results). The better performance in our tests may be due to our implementation of LSH; we use Charikar's random projection method [1] to construct hash tables.

In terms of training time, the BRE method typically converges in 50–100 iterations of updating all hash functions, and takes 1–5 minutes to train per data set on our machines (depending on the number of bits requested). Relatively speaking, the time required for training is typically faster than RBM but slower than spectral hashing and LSH. Search times in the binary space are uniform across each of the methods and our timing results are similar to those reported previously (see, e.g. [17]).

## 3.3 Qualitative Results

Finally, we present qualitative results on the large Tiny Image data set [5] to demonstrate our method applied to a very large database. This data set contains 80 million images, and is one of the largest readily available data sets for content-based image retrieval. Each image is stored as $32 \times 32$ pixels, and we employ the global Gist descriptors that have been extracted for each image.

We ran our reconstructive hashing algorithm on the Gist descriptors for the Tiny Image data set using 50 bits, with 1000 training images used to construct the hash functions as before. We selected a random set of queries from the database and compared the results of a linear scan over the Gist features with the hashing results over the Gist features. When obtaining hashing results, we collected the nearest neighbors in the Hamming space to the query (the top $0.01\%$ of the Hamming distances), and then sorted these by their distance in the original Gist space. Some example results are displayed in Figure 2; we see that, with 50 bits, we can obtain very good results that are qualitatively similar to the results of the linear scan.

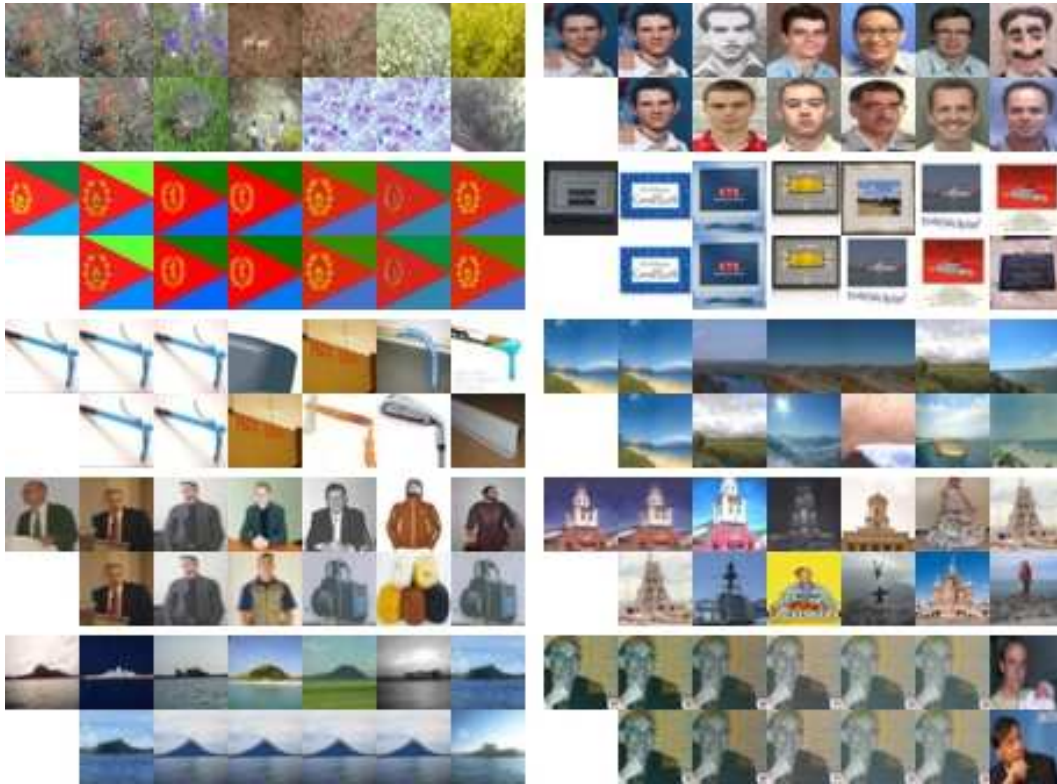

Figure 2: Qualitative results over the 80 million images in the Tiny Image database [5]. For each group of images, the top left image is the query, the top row corresponds to a linear scan, and the second row corresponds to the hashing retrieval results using 50 hash bits. The hashing results are similar to the linear scan results but are significantly faster to obtain.

## 4 Conclusion and Future Work

In this paper, we presented a method for learning hash functions, developed an efficient coordinate-descent algorithm for finding a local optimum, and demonstrated improved performance on several benchmark vision data sets as compared to existing state-of-the-art hashing algorithms. One avenue for future work is to explore alternate methods of optimization; our approach, while simple and fast, may fall into poor local optima in some cases. Second, we would like to explore the use of our algorithm in the supervised setting for large-scale $k$-NN tasks.

**Acknowledgments**

This work was supported in part by DARPA, Google, and NSF grants IIS-0905647 and IIS-0819984. We thank Rob Fergus for the spectral hashing and RBM code, and Greg Shakhnarovich for the Boosting SSC code.

## Footnotes

[1]Alternatively, we may scale the data appropriately by a constant so that the squared Euclidean distances $\frac{1}{2}\|\boldsymbol{x}_i - \boldsymbol{x}_j\|^2$ are in $[0, 1]$.

[2]The sigmoid function is defined as $s(x) = 1/(1 + e^{-x})$, and its derivative is $s'(x) = s(x)(1 - s(x))$.

## References

[1] M. Charikar. Similarity Estimation Techniques from Rounding Algorithms. In *STOC*, 2002.

[2] P. Indyk and R. Motwani. Approximate Nearest Neighbors: Towards Removing the Curse of Dimensionality. In *STOC*, 1998.

[3] R. R. Salakhutdinov and G. E. Hinton. Learning a Nonlinear Embedding by Preserving Class Neighbourhood Structure. In *AISTATS*, 2007.

[4] Y. Weiss, A. Torralba, and R. Fergus. Spectral Hashing. In *NIPS*, 2008.

[5] A. Torralba, R. Fergus, and W. T. Freeman. 80 Million Tiny Images: A Large Dataset for Non-parametric Object and Scene Recognition. *TPAMI*, 30(11):1958–1970, 2008.

[6] J. Freidman, J. Bentley, and A. Finkel. An Algorithm for Finding Best Matches in Logarithmic Expected Time. *ACM Transactions on Mathematical Software*, 3(3):209–226, September 1977.

[7] P. Ciaccia, M. Patella, and P. Zezula. M-tree: An Efficient Access Method for Similarity Search in Metric Spaces. In *VLDB*, 1997.

[8] A. Beygelzimer, S. Kakade, and J. Langford. Cover Trees for Nearest Neighbor. In *ICML*, 2006.

[9] J. Uhlmann. Satisfying General Proximity / Similarity Queries with Metric Trees. *Information Processing Letters*, 40:175–179, 1991.

[10] M. Datar, N. Immorlica, P. Indyk, and V. Mirrokni. Locality-Sensitive Hashing Scheme Based on p-Stable Distributions. In *SOCG*, 2004.

[11] P. Jain, B. Kulis, and K. Grauman. Fast Image Search for Learned Metrics. In *CVPR*, 2008.

[12] K. Grauman and T. Darrell. Pyramid Match Hashing: Sub-Linear Time Indexing Over Partial Correspondences. In *CVPR*, 2007.

[13] G. Shakhnarovich, P. Viola, and T. Darrell. Fast Pose Estimation with Parameter-Sensitive Hashing. In *ICCV*, 2003.

[14] G. Shakhnarovich. *Learning Task-specific Similarity*. PhD thesis, MIT, 2006.

[15] N. Snavely, S. Seitz, and R. Szeliski. Photo Tourism: Exploring Photo Collections in 3D. In *SIGGRAPH Conference Proceedings*, pages 835–846, New York, NY, USA, 2006. ACM Press.

[16] L. Fei-Fei, R. Fergus, and P. Perona. Learning Generative Visual Models from Few Training Examples: an Incremental Bayesian Approach Tested on 101 Object Categories. In *Workshop on Generative Model Based Vision*, Washington, D.C., June 2004.

[17] A. Torralba, R. Fergus, and Y. Weiss. Small Codes and Large Databases for Recognition. In *CVPR*, 2008.

